# Approximate Correspondences in High Dimensions

**Kristen Grauman**
Department of Computer Sciences
University of Texas at Austin
grauman@cs.utexas.edu

**Trevor Darrell**
CS and AI Laboratory
Massachusetts Institute of Technology
trevor@csail.mit.edu

## Abstract

Pyramid intersection is an efficient method for computing an approximate partial matching between two sets of feature vectors. We introduce a novel pyramid embedding based on a hierarchy of non-uniformly shaped bins that takes advantage of the underlying structure of the feature space and remains accurate even for sets with high-dimensional feature vectors. The matching similarity is computed in linear time and forms a Mercer kernel. Whereas previous matching approximation algorithms suffer from distortion factors that increase linearly with the feature dimension, we demonstrate that our approach can maintain constant accuracy even as the feature dimension increases. When used as a kernel in a discriminative classifier, our approach achieves improved object recognition results over a state-of-the-art set kernel.

## 1   Introduction

When a single data object is described by a set of feature vectors, it is often useful to consider the matching or "correspondence" between two sets' elements in order to measure their overall similarity or recover the alignment of their parts. For example, in computer vision, images are often represented as collections of local part descriptions extracted from regions or patches (e.g., [11, 12]), and many recognition algorithms rely on establishing the correspondence between the parts from two images to quantify similarity between objects or localize an object within the image [2, 3, 7]. Likewise, in text processing, a document may be represented as a bag of word-feature vectors; for example, Latent Semantic Analysis can be used to recover a "word meaning" subspace on which to project the co-occurrence count vectors for every word [9]. The relationship between documents may then be judged in terms of the matching between the sets of local meaning features.

The critical challenge, however, is to compute the correspondences between the feature sets in an efficient way. The optimal correspondences—those that minimize the matching cost—require cubic time to compute, which quickly becomes prohibitive for sizeable sets and makes processing realistic large data sets impractical. Due to the optimal matching's complexity, researchers have developed approximation algorithms to compute close solutions for a fraction of the computational cost [4, 8, 1, 7]. However, previous approximations suffer from distortion factors that increase linearly with the dimension of the features, and they fail to take advantage of structure in the feature space.

In this paper we present a new algorithm for computing an approximate partial matching between point sets that can remain accurate even for sets with high-dimensional feature vectors, and benefits from taking advantage of the underlying structure in the feature space. The main idea is to derive a hierarchical, data-dependent decomposition of the feature space that can be used to encode feature sets as multi-resolution histograms with non-uniformly shaped bins. For two such histograms (*pyramids*), the matching cost is efficiently calculated by counting the number of features that intersect in each bin, and weighting these match counts according to geometric estimates of inter-feature distances. Our method allows for partial matchings, which means that the input sets can have varying numbers of features in them, and outlier features from the larger set can be ignored with no penalty

to the matching cost. The matching score is computed in time linear in the number of features per set, and it forms a Mercer kernel suitable for use within existing kernel-based algorithms.

In this paper we demonstrate how, unlike previous set matching approximations (including our original pyramid match algorithm [7]), the proposed approach can maintain consistent accuracy as the dimension of the features within the sets increases. We also show how the data-dependent hierarchical decomposition of the feature space produces more accurate correspondence fields than a previous approximation that uses a uniform decomposition. Finally, using our matching measure as a kernel in a discriminative classifier, we achieve improved object recognition results over a state-of-the-art set kernel on a benchmark data set.

## 2 Related Work

Several previous matching approximation methods have also considered a hierarchical decomposition of the feature space to reduce matching complexity, but all suffer from distortion factors that scale linearly with the feature dimension [4, 8, 1, 7]. In this work we show how to alleviate this decline in accuracy for high-dimensional data by tuning the hierarchical decomposition according to the particular structure of the data, when such structure exists.

We build on our pyramid match algorithm [7], a partial matching approximation that also uses histogram intersection to efficiently count matches implicitly formed by the bin structures. However, in contrast to [7], our use of data-dependent, non-uniform bins and a more precise weighting scheme results in matchings that are consistently accurate for structured, high-dimensional data.

The idea of partitioning a feature space with vector quantization (VQ) is fairly widely used in practice; in the vision literature in particular, VQ has been used to establish a vocabulary of prototypical image features, from "textons" to the "visual words" of [16]. A variant of the pyramid match applied to spatial features was shown to be effective for matching quantized features in [10]. More recently, the authors of [13] have shown that a tree-structured vector quantization (TSVQ [5]) of image features provides a scalable means of indexing into a very large feature vocabulary. The actual tree structure employed is similar to the one constructed in this work; however, whereas the authors of [13] are interested in matching individual features to one another to access an inverted file, our approach computes approximate correspondences between *sets* of features. Note the distinction between the problem we are addressing—approximate matchings between sets—and the problem of efficiently identifying approximate or exact nearest neighbor feature vectors (e.g., via $k$-$d$ trees): in the former, the goal is a one-to-one correspondence between sets of vectors, whereas in the latter, a single vector is independently matched to a nearby vector.

## 3 Approach

The main contribution of this work is a new very efficient approximate bipartite matching method that measures the correspondence-based similarity between unordered, variable-sized sets of vectors, and can optionally extract an explicit correspondence field. We call our algorithm the *vocabulary-guided* (VG) *pyramid match*, since the histogram pyramids are defined by the "vocabulary" or structure of the feature space, and the pyramids are used to count implicit matches.

The basic idea is to first partition the given feature space into a pyramid of non-uniformly shaped regions based on the distribution of a provided corpus of feature vectors. Point sets are then encoded as multi-resolution histograms determined by that pyramid, and an efficient intersection-based computation between any two histogram pyramids yields an approximate matching score for the original sets. The implicit matching version of our method estimates the inter-feature distances based on their respective distances to the bin centers. To produce an explicit correspondence field between the sets, we use the pyramid construct to divide-and-conquer the optimal matching computation. As our experiments will show, the proposed algorithm in practice provides a good approximation to the optimal partial matching, but is orders of magnitude faster to compute.

**Preliminaries:** We consider a feature space $F$ of $d$-dimensional vectors, $F \subseteq \Re^d$. The point sets our algorithm matches will come from the input space $S$, which contains sets of feature vectors drawn from $F$: $S = \{\mathbf{X} | \mathbf{X} = \{\mathbf{x}_1, \ldots, \mathbf{x}_m\}\}$, where each $\mathbf{x}_i \in F$, and the value $m = |\mathbf{X}|$ may vary across instances of sets in $S$. Throughout the text we will use the terms feature, vector, and point interchangeably to refer to the elements within a set.

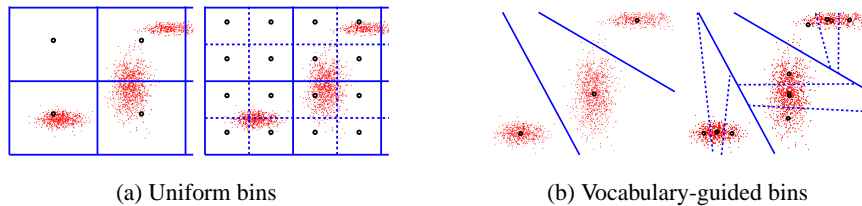

| (a) Uniform bins | (b) Vocabulary-guided bins |
|---|---|

Figure 1: Rather than carve the feature space into uniformly-shaped partitions (left), we let the vocabulary (structure) of the feature space determine the partitions (right). As a result, the bins are better concentrated on decomposing the space where features cluster, particularly for high-dimensional feature spaces. These figures depict the grid boundaries for two resolution levels for a 2-D feature space. In both (a) and (b), the left plot contains the coarser resolution level, and the right plot contains the finer one. Features are red points, bin centers are larger black points, and blue lines denote bin boundaries.

A partial matching between two point sets is an assignment that maps all points in the smaller set to some subset of the points in the larger (or equally-sized) set. Given point sets $\mathbf{X}$ and $\mathbf{Y}$, where $m = |\mathbf{X}|$, $n = |\mathbf{Y}|$, and $m \leq n$, a partial matching $\mathcal{M}(\mathbf{X}, \mathbf{Y}; \pi) = \{(\mathbf{x}_1, \mathbf{y}_{\pi_1}), \ldots, (\mathbf{x}_m, \mathbf{y}_{\pi_m})\}$ pairs each point in $\mathbf{X}$ to some unique point in $\mathbf{Y}$ according to the permutation of indices specified by $\pi = [\pi_1, \ldots, \pi_m]$, $1 \leq \pi_i \leq n$, where $\pi_i$ specifies which point $\mathbf{y}_{\pi_i} \in \mathbf{Y}$ is matched to $\mathbf{x}_i \in \mathbf{X}$, for $1 \leq i \leq m$. The cost of a partial matching is the sum of the distances between matched points: $\mathcal{C}(\mathcal{M}(\mathbf{X}, \mathbf{Y}; \pi)) = \sum_{\mathbf{x}_i \in \mathbf{X}} ||\mathbf{x}_i - \mathbf{y}_{\pi_i}||_2$. The optimal partial matching $\mathcal{M}(\mathbf{X}, \mathbf{Y}; \pi^*)$ uses the assignment $\pi^*$ that minimizes this cost: $\pi^* = \mathrm{argmin}_\pi \mathcal{C}(\mathcal{M}(\mathbf{X}, \mathbf{Y}; \pi))$. It is this matching that we wish to efficiently approximate. In Section 3.2 we describe how our algorithm approximates the cost $\mathcal{C}(\mathcal{M}(\mathbf{X}, \mathbf{Y}; \pi^*))$; for a small increase in computational cost we can also extract explicit correspondences to estimate $\pi^*$ itself.

## 3.1 Building Vocabulary-Guided Pyramids

The first step is to generate the structure of the vocabulary-guided (VG) pyramid to define the bin placement for the multi-resolution histograms used in the matching. This is a one-time process performed before any matching takes place. We would like the bins in the pyramid to follow the feature distribution and concentrate partitions where the features actually fall. To accomplish this, we perform hierarchical clustering on a sample of representative feature vectors from $F$.

We randomly select some example feature vectors from the feature type of interest to form the representative feature corpus, and perform hierarchical $k$-means clustering with the Euclidean distance to build the pyramid tree. Other hierarchical clustering techniques, such as agglomerative clustering, are also possible and do not change the operation of the method. For this unsupervised clustering process there are two parameters: the number of levels in the tree $L$, and the branching factor $k$. The initial corpus of features is clustered into $k$ top-level groups, where group membership is determined by the Voronoi partitioning of the feature corpus according to the $k$ cluster centers. Then the clustering is repeated recursively $L - 1$ times on each of these groups, filling out a tree with $L$ total levels containing $k^i$ bins (nodes) at level $i$, where levels are counted from the root ($i = 0$) to the leaves ($i = L - 1$). The bins are irregularly shaped and sized, and their boundaries are determined by the Voronoi cells surrounding the cluster centers. (See Figure 1.) For each bin in the VG pyramid we record its diameter, which we estimate empirically based on the maximal inter-feature distance between any points from the initial feature corpus that were assigned to it.

Once we have constructed a VG pyramid, we can embed point sets from $S$ as multi-resolution histograms. A point's placement in the pyramid is determined by comparing it to the appropriate $k$ bin centers at each of the $L$ pyramid levels. The histogram count is incremented for the bin (among the $k$ choices) that the point is nearest to in terms of the same distance function used to cluster the initial corpus. We then push the point down the tree and continue to increment finer level counts only along the branch (bin center) that is chosen at each level. So a point is first assigned to one of the top-level clusters, then it is assigned to one of *its* children, and so on recursively. This amounts to a total of $kL$ distances that must be computed between a point and the pyramid's bin centers.

Given the bin structure of the VG pyramid, a point set $\mathbf{X}$ is mapped to its pyramid: $\Psi(\mathbf{X}) = [H_0(\mathbf{X}), \ldots, H_{L-1}(\mathbf{X})]$, with $H_i(\mathbf{X}) = [\langle \mathbf{p}, n, d \rangle_1, \ldots, \langle \mathbf{p}, n, d \rangle_{k^i}]$, and where $H_i(\mathbf{X})$ is a $k^i$-dimensional histogram associated with level $i$ in the pyramid, $\mathbf{p} \in \mathbb{Z}^i$ for entries in $H_i(\mathbf{X})$, and

$0 \leq i < L$. Each entry in this histogram is a triple $\langle \mathbf{p}, n, d \rangle$ giving the bin index, the bin count, and the bin's points' maximal distance to the bin center, respectively.

Storing the VG pyramid itself requires space for $O(k^L)$ $d$-dimensional feature vectors, i.e., all of the cluster centers. However, each point set's histogram is stored sparsely, meaning only $O(mL)$ nonzero bin counts are maintained to encode the entire pyramid for a set with $m$ features. This is an important point: we do not store $O(k^L)$ counts for every point set; $H_i(\mathbf{X})$ is represented by at most $m$ triples having $n > 0$. We achieve a sparse implementation as follows: each vector in a set is pushed through the tree as described above. At every level $i$, we record a $\langle \mathbf{p}, n, d \rangle$ triple describing the nonzero entry for the current bin. The vector $\mathbf{p} = [p_1, \ldots, p_i]$, $p_j \in [1, k]$ denotes the indices of the clusters traversed from the root so far, $n \in \mathbb{Z}^+$ denotes the count for the bin (initially 1), and $d \in \Re$ denotes the distance computed between the inserted point and the current bin's center. Upon reaching the leaf level, $\mathbf{p}$ is an $L$-dimensional path-vector indicating which of the $k$ bins were chosen at each level, and every path-vector uniquely identifies some bin on the pyramid.

Initially, an input set with $m$ features yields a total of $mL$ such triples—there is one nonzero entry per level per point, and each has $n = 1$. Then each of the $L$ lists of entries is sorted by the index vectors ($\mathbf{p}$ in the triple), and they are collapsed to a list of sorted nonzero entries with unique indices: when two or more entries with the same index are found, they are replaced with a single entry with the same index for $\mathbf{p}$, the summed counts for $n$, and the maximum distance for $d$. The sorting is done in linear time using integer sorting algorithms. Maintaining the maximum distance of any point in a bin to the bin center will allow us to efficiently estimate inter-point distances at the time of matching, as described in Section 3.2.

## 3.2 Vocabulary-Guided Pyramid Match

Given two point sets' pyramid encodings, we efficiently compute the approximate matching score using a simple weighted intersection measure. The VG pyramid's multi-resolution partitioning of the feature space is used to direct the matching. The basic intuition is to start collecting groups of matched points from the bottom of the pyramid up, i.e., from within increasingly larger partitions. In this way, we will first consider matching the closest points (at the leaves), and as we climb to the higher-level clusters in the pyramid we will allow increasingly further points to be matched. We define the number of *new* matches within a bin to be a count of the minimum number of points either of the two input sets contributes to that bin, minus the number of matches already counted by any of its child bins. A weighted sum of these counts yields an approximate matching score.

Let $n_{ij}(\mathbf{X})$ denote the element $n$ from $\langle \mathbf{p}, n, d \rangle_j$, the $j^{th}$ bin entry of histogram $H_i(\mathbf{X})$, and let $c_h (n_{ij}(\mathbf{X}))$ denote the element $n$ for the $h^{th}$ child bin of that entry, $1 \leq h \leq k$. Similarly, let $d_{ij}(\mathbf{X})$ refer to the element $d$ from the same triple. Given point sets $\mathbf{X}$ and $\mathbf{Y}$, we compute the matching score via their pyramids $\Psi(\mathbf{X})$ and $\Psi(\mathbf{Y})$ as follows:

$$\mathcal{C}\left(\Psi(\mathbf{X}), \Psi(\mathbf{Y})\right) = \sum_{i=0}^{L-1} \sum_{j=1}^{k^i} w_{ij} \left[ \min\left(n_{ij}(\mathbf{X}), n_{ij}(\mathbf{Y})\right) - \sum_{h=1}^{k} \min\left(c_h\left(n_{ij}(\mathbf{X})\right), c_h\left(n_{ij}(\mathbf{Y})\right)\right) \right]. \quad (1)$$

The outer sum loops over the levels in the pyramids; the second sum loops over the bins at a given level, and the innermost sum loops over the children of a given bin. The first $\min$ term reflects the number of matchable points in the current bin, and the second $\min$ term tallies the number of matches already counted at finer resolutions (in child bins). Note that as the leaf nodes have no children, when $i = L - 1$ the last sum is zero. All matches are new at the leaves. The matching scores are normalized according to the size of the input sets in order to not favor larger sets.

The number of new matches calculated for a bin is weighted by $w_{ij}$, an estimate of the distance between points contained in the bin.[1] With a VG pyramid match there are two alternatives for the distance estimate: (a) weights based on the diameters of the pyramid's bins, or (b) input-dependent weights based on the maximal distances of the points in the bin to its center. Option (a) is a conservative estimate of the actual inter-point distances in the bin if the corpus of features used to build the pyramid is representative of the feature space; its advantages are that it provides a guaranteed Mercer kernel (see below) and eliminates the need to store a distance $d$ in the entry triples. Option (b)'s input-specific weights estimate the distance between any two points in the bin as the sum of the stored maximal to-center distances from either input set: $w_{ij} = d_{ij}(\mathbf{X}) + d_{ij}(\mathbf{Y})$. This weighting

gives a true upper bound on the furthest any two points could be from one another, and it has the potential to provide tighter estimates of inter-feature distances (as we confirm experimentally below); however, we cannot guarantee this weighting will yield a Mercer kernel.

Just as we encode the pyramids sparsely, we derive a means to compute intersections in Eqn. 1 without ever traversing the entire pyramid tree. Given two sparse lists $H_i(\mathbf{X})$ and $H_i(\mathbf{Y})$ which have been sorted according to the bin indices, we obtain the minimum counts in linear time by moving pointers down the lists and processing only those nonzero entries that share an index, making the time required to compute a matching between two pyramids $O(mL)$. A key aspect of our method is that we obtain a measure of matching quality between two point sets without computing pair-wise distances between their features—an $O(m^2)$ savings over sub-optimal greedy matchings. Instead, we exploit the fact that the points' placement in the pyramid reflects their distance from one another. The only inter-feature distances computed are the $kL$ distances needed to insert a point into the pyramid, and this small one-time cost is amortized every time we re-use a histogram to approximate another matching against a different point set.

We first suggested the idea of using histogram intersection to count implicit matches in a multi-resolution grid in [7]. However, in [7], bins are constructed to uniformly partition the space, bin diameters exponentially increase over the levels, and intersections are weighted indistinguishably across an entire level. In contrast, here we have developed a pyramid embedding that partitions according to the distribution of features, and weighting schemes that allow more precise approximations of the inter-feature costs. As we will show in Section 4, our VG pyramid match remains accurate and efficient even for high-dimensional feature spaces, while the uniform-bin pyramid match is limited in practice to relatively low-dimensional features.

For the increased accuracy our method provides, there are some complexity trade-offs versus [7], which does not require computing any distances to place the points into bins; their uniform shape and size allows points to be placed directly via division by bin size. On the other hand, sorting the bin indices with the VG method has a lower complexity, since the values only range to $k$, the branch factor, which is typically much smaller than the aspect ratio that bounds the range in [7]. In addition, as we show in Section 4, in practice the cost of extracting an *explicit* correspondence field using the uniform-bin pyramid in high dimensions approaches the cubic cost of the optimal measure, whereas it remains linear with the proposed approach, assuming features are not uniformly distributed.

Our approximation can be used to compare sets of vectors in any case where the presence of low-cost correspondences indicates their similarity (e.g., nearest-neighbor retrieval). We can also employ the measure as a kernel function for structured inputs. According to Mercer's theorem, a kernel is p.s.d if and only if it corresponds to an inner product in some feature space [15]. We can re-write Eqn. 1 as: $\mathcal{C}\left(\Psi(\mathbf{X}), \Psi(\mathbf{Y})\right) = \sum_{i=0}^{L-1} \sum_{j=1}^{k^i} \left(w_{ij} - p_{ij}\right) \min\left(n_{ij}(\mathbf{X}), n_{ij}(\mathbf{Y})\right)$, where $p_{ij}$ refers to the weight associated with the parent bin of the $j^{th}$ node at level $i$. Since the $\min$ operation is p.d. [14], and since kernels are closed under summation and scaling by a positive constant [15], we have that the VG pyramid match is a Mercer kernel if $w_{ij} \geq p_{ij}$. This inequality holds if every child bin receives a similarity weight that is greater than its parent bin, or rather that every child bin has a distance estimate that is less than that of its parent. Indeed this is the case for weighting option (a), where $w_{ij}$ is inversely proportional to the diameter of the bin. It holds by definition of the hierarchical clustering: the diameter of a subset of points must be less than or equal to the diameter of *all* those points. We cannot make this guarantee for weighting option (b).

In addition to scalar matching scores, we can optionally extract explicit correspondence fields through the pyramid. In this case, the VG pyramid decomposes the required matching computation into a hierarchy of smaller matchings. Upon encountering a bin with a nonzero intersection, the optimal matching is computed between only those features from the two sets that fall into that particular bin. All points that are used in that per-bin matching are then flagged as matched and may not take part in subsequent matchings at coarser resolutions of the pyramid.

## 4   Results

In this section, we provide results to empirically demonstrate our matching's accuracy and efficiency on real data, and we compare it to a pyramid match using a uniform partitioning of the feature space. In addition to directly evaluating the matching scores and correspondence fields, we show that our method leads to improved object recognition performance when used as a kernel within a discriminative classifier.

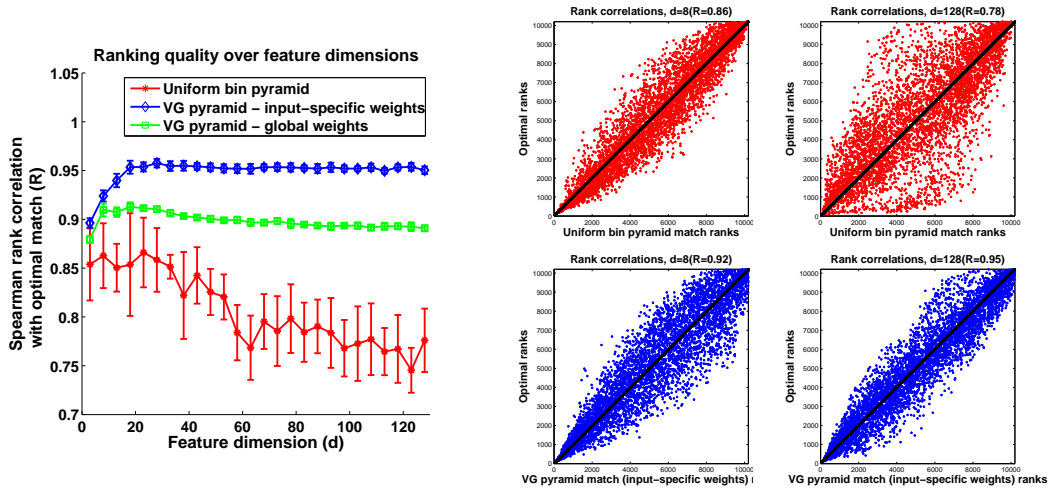

Figure 2: Comparison of optimal and approximate matching rankings on image data. *Left:* The set rankings produced with the VG pyramid match are consistently accurate for increasing feature dimensions, while the accuracy with uniform bins degrades about linearly in the feature dimension. *Right:* Example rankings for both approximations at $d = [8, 128]$.

**Approximate Matching Scores:** In these experiments, we extracted local SIFT [11] features from images in the ETH-80 database, producing an unordered set of about $m = 256$ vectors for every example. In this case, $F$ is the space of SIFT image features. We sampled some features from 300 of the images to build the VG pyramid, and 100 images were used to test the matching. In order to test across varying feature dimensions, we also used some training features to establish a PCA subspace that was used to project features onto varying numbers of bases. For each feature dimension, we built a VG pyramid with $k = 10$ and $L = 5$, encoded the 100 point sets as pyramids, and computed the pair-wise matching scores with both our method and the optimal least-cost matching.

If our measure is approximating the optimal matching well, we should find the ranking we induce to be highly correlated with the ranking produced by the optimal matching for the same data. In other words, the images should be sorted similarly by either method. Spearman's rank correlation coefficient $R$ provides a good quantitative measure to evaluate this: $R = 1 - 6 \sum_1^N D^2 / N(N^2 - 1)$, where $D$ is the difference in rank for the $N$ corresponding ordinal values assigned by the two measures. The left plot in Figure 2 shows the Spearman correlation scores against the optimal measure for both our method (with both weighting options) and the approximation in [7] for varying feature dimensions for the 10,000 pair-wise matching scores for the 100 test sets. Due to the randomized elements of the algorithms, for each method we have plotted the mean and standard deviation of the correlation for 10 runs on the same data.

While the VG pyramid match remains consistently accurate for high feature dimensions ($R = 0.95$ with input-specific weights), the accuracy of the uniform bins degrades rapidly for dimensions over 10. The ranking quality of the input-specific weighting scheme (blue diamonds) is somewhat stronger than that of the "global" bin diameter weighting scheme (green squares). The four plots on the right of Figure 2 display the actual ranks computed for both approximations for two of the 26 dimensions summarized in the left plot. The black diagonals denote the optimal performance, where the approximate rankings would be identical to the optimal ones; higher Spearman correlations have points clustered more tightly along this diagonal. For the low-dimensional features, the methods perform fairly comparably; however, for the full 128-D features, the VG pyramid match is far superior (rightmost column). The optimal measure requires about $1.25s$ per match, while our approximation is about $2500x$ faster at $5 \times 10^{-4}s$ per match. Computing the pyramid structure from the feature corpus took about three minutes in Matlab; this is a one-time offline cost.

For a pyramid matching to work well, the gradation in bin sizes up the pyramid must be such that at most levels of the pyramid we can capture distinct groups of points to match within the bins. That is, unless all the points in two sets are equidistant, the bin placement must allow us to match very near points at the finest resolutions, and gradually add matches that are more distant at coarser resolutions. In low dimensions, both uniform or data-dependent bins can achieve this. In high dimensions, however, uniform bin placement and exponentially increasing bin diameters fail to capture such a gradation: once any features from different point sets are close enough to

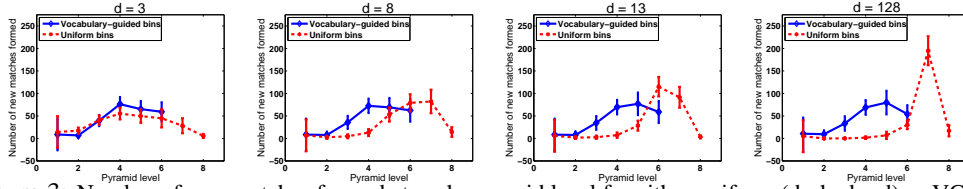

Figure 3: Number of *new* matches formed at each pyramid level for either uniform (dashed red) or VG (solid blue) bins for increasing feature dimensions. Points represent mean counts per level for 10,000 matches. In low dimensions, both partition styles gradually collect matches up the pyramid. In high dimensions with uniform partitions, points begin sharing a bin "all at once"; in contrast, the VG bins still accrue new matches consistently across levels since the decomposition is tailored to where points cluster in the feature space.

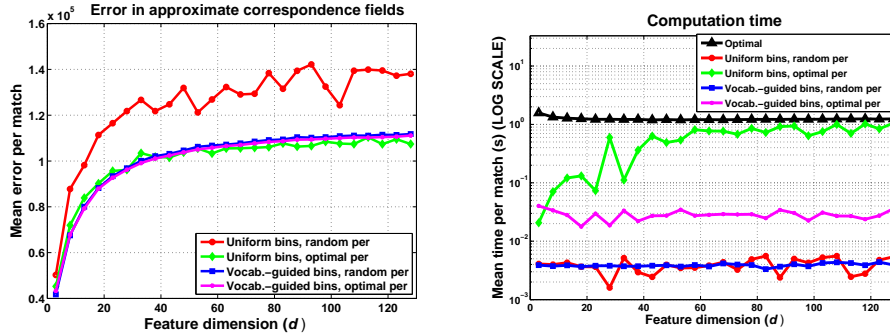

Figure 4: Comparison of correspondence field errors (left) and associated computation times (right). This figure is best viewed in color. (Note that errors level out with $d$ for all methods due to PCA.)

match (share bins), the bins are so large that almost *all* of them match. The matching score is then approximately the number of points weighted by a single bin size. In contrast, when we tailor the feature space partitions to the distribution of the data, even in high dimensions the match counts increase gradually across levels, thereby yielding more discriminating implicit matches. Figure 3 confirms this intuition, again using the ETH-80 image data from above.

**Approximate Correspondence Fields:** For the same image data, we ran the explicit matching variant of our method and compared the induced correspondences to those produced by the globally optimal measure. For comparison, we also applied the same variant to pyramids with uniform bins. We measure the error of an approximate matching $\hat{\pi}$ by the sum of the errors at every link in the field: $E\left(\mathcal{M}\left(\mathbf{X}, \mathbf{Y}; \hat{\pi}\right), \mathcal{M}\left(\mathbf{X}, \mathbf{Y}; \pi^*\right)\right) = \sum_{\mathbf{x}_i \in \mathbf{X}} ||\mathbf{y}_{\hat{\pi}_i} - \mathbf{y}_{\pi_i^*}||_2$. Figure 4 compares the correspondence field error and computation times for the VG and uniform pyramids. For each approximation, there are two variations tested: in one, an optimal assignment is computed for all points in the same bin; for the other, a random assignment is made. The left plot shows the mean error per match for each method, and the right plot shows the corresponding mean time required to compute those matches.

The computation times are as we would expect: the optimal matching is orders of magnitude more expensive than the approximations. Using the random assignment variation, both approximations have negligible costs, since they simply choose any combination of points within a bin. However, in high dimensions, the time required by the uniform bin pyramid with the optimal per-bin matching approaches the time required by the optimal matching itself. This occurs for similar reasons as the poorer matching score accuracy exhibited by the uniform bins, both in the left plot and above in Figure 2; since most or all of the points begin to match at a certain level, the pyramid does not help to divide-and-conquer the computation, and for high dimensions, the optimal matching in its entirety must be computed. In contrast, the expense of the VG pyramid matching remains steady and low, even for high dimensions, since data-dependent pyramids better divide the matching labor into the natural segments in the feature space.

For similar reasons, the errors are comparable for the optimal per-bin variation with either the VG or uniform bins. The VG bins divide the computation so it can be done inexpensively, while the uniform bins divide the computation poorly and must compute it expensively, but about as accurately. Likewise, the error for the uniform bins when using a per-bin random assignment is very high for any but the lowest dimensions (red line on left plot), since such a large number of points are being randomly assigned to one another. In contrast, the VG bins actually result in similar errors whether the points in a bin are matched optimally or randomly (blue and pink lines on left plot).

| Pyramid matching method | Mean recognition rate/class ($d$=128 / $d$=10) | Time/match (s) ($d$=128 / $d$=10) |
|---|---|---|
| Vocabulary-guided bins | **99.0 / 97.7** | 6.1e-4 / 6.2e-4 |
| Uniform bins | 64.9 / 96.5 | 1.5e-3 / 5.7e-4 |

This again indicates that tuning the pyramid bins to the data's distribution achieves a much more suitable breakdown of the computation, even in high dimensions.

**Realizing Improvements in Recognition:** Finally, we have experimented with the VG pyramid match within a discriminative classifier for an object recognition task. We trained an SVM with our matching as the kernel to recognize the four categories in the Caltech-4 benchmark data set. We trained with 200 images per class and tested with all the remaining images. We extracted features using both the Harris and MSER [12] detectors and the 128-D SIFT [11] descriptor. We also generated lower-dimensional ($d = 10$) features using PCA. To form a Mercer kernel, the weights were set according to each bin diameter $A_{ij}$: $w_{ij} = e^{-A_{ij}/\sigma}$, with $\sigma$ set automatically as the mean distance between a sample of features from the training set. The table shows our improvements over the uniform-bin pyramid match kernel. The VG pyramid match is more accurate and requires minor additional computation. Our near-perfect performance on this data set is comparable to that reached by others in the literature; the real significance of the result is that it distinguishes what can be achieved with a VG pyramid embedding as opposed to the uniform histograms used in [7], particularly for high-dimensional features. In addition, here the optimal matching requires 0.31$s$ per match, over 500$x$ the cost of our method.

**Conclusion:** We have introduced a linear-time method to compute a matching between point sets that takes advantage of the underlying structure in the feature space and remains consistently accurate and efficient for high-dimensional inputs on real image data. Our results demonstrate the strength of the approximation empirically, compare it directly against an alternative state-of-the-art approximation, and successfully use it as a Mercer kernel for an object recognition task. We have commented most on potential applications in vision and text, but in fact it is a generic matching measure that can be applied whenever it is meaningful to compare sets by their correspondence.

**Acknowledgments:** We thank Ben Kuipers for suggesting the use of Spearman's rank correlation.

## Footnotes

[1] To use our matching as a cost function, weights are set as the distance estimates; to use as a similarity measure or kernel, weights are set as (some function of) the inverse of the distance estimates.

# References

[1] P. Agarwal and K. R. Varadarajan. A Near-Linear Algorithm for Euclidean Bipartite Matching. In *Symposium on Computational Geometry*, 2004.

[2] S. Belongie, J. Malik, and J. Puzicha. Shape Matching and Object Recognition Using Shape Contexts. *IEEE Trans. on Pattern Analysis and Machine Intelligence*, 24(24):509–522, April 2002.

[3] A. Berg, T. Berg, and J. Malik. Shape Matching and Object Recognition using Low Distortion Correspondences. In *Proc. IEEE Conf. on Comp. Vision and Pattern Recognition*, San Diego, CA, June 2005.

[4] M. Charikar. Similarity Estimation Techniques from Rounding Algorithms. In *Proceedings of the 34th Annual ACM Symposium on Theory of Computing*, 2002.

[5] A. Gersho and R. Gray. *Vector Quantization and Signal Compression*. Springer, 1992.

[6] K. Grauman. *Matching Sets of Features for Efficient Retrieval and Recognition*. PhD thesis, MIT, 2006.

[7] K. Grauman and T. Darrell. The Pyramid Match Kernel: Discriminative Classification with Sets of Image Features. In *Proc. IEEE Int. Conf. on Computer Vision*, Beijing, China, Oct 2005.

[8] P. Indyk and N. Thaper. Fast Image Retrieval via Embeddings. In *3rd International Workshop on Statistical and Computational Theories of Vision*, Nice, France, Oct 2003.

[9] T. K. Landauer, P. W. Foltz, and D. Laham. Introduction to LSA. *Discourse Processes*, 25:259–84, 1998.

[10] S. Lazebnik, C. Schmid, and J. Ponce. Beyond Bags of Features: Spatial Pyramid Matching for Recognizing Scene Categories. In *Proc. IEEE Conf. on Comp. Vision and Pattern Recognition*, June 2006.

[11] D. Lowe. Distinctive Image Features from Scale-Invariant Keypoints. *International Journal of Computer Vision*, 60(2):91–110, Jan 2004.

[12] J. Matas, O. Chum, M. Urban, and T. Pajdla. Robust Wide Baseline Stereo from Maximally Stable Extremal Regions. In *British Machine Vision Conference*, Cardiff, UK, Sept. 2002.

[13] D. Nister and H. Stewenius. Scalable Recognition with a Vocabulary Tree. In *Proc. IEEE Conf. on Comp. Vision and Pattern Recognition*, New York City, NY, June 2006.

[14] F. Odone, A. Barla, and A. Verri. Building Kernels from Binary Strings for Image Matching. *IEEE Trans. on Image Processing*, 14(2):169–180, Feb 2005.

[15] J. Shawe-Taylor and N. Cristianini. *Kernel Methods for Pattern Analysis*. Cambridge Univ. Press, 2004.

[16] J. Sivic and A. Zisserman. Video Google: A Text Retrieval Approach to Object Matching in Videos. In *Proc. IEEE Int. Conf. on Computer Vision*, Nice, Oct 2003.
